# The Cascade-Correlation Learning Architecture

**Scott E. Fahlman and Christian Lebiere**
School of Computer Science
Carnegie-Mellon University
Pittsburgh, PA 15213

## ABSTRACT

Cascade-Correlation is a new architecture and supervised learning algorithm for artificial neural networks. Instead of just adjusting the weights in a network of fixed topology, Cascade-Correlation begins with a minimal network, then automatically trains and adds new hidden units one by one, creating a multi-layer structure. Once a new hidden unit has been added to the network, its input-side weights are frozen. This unit then becomes a permanent feature-detector in the network, available for producing outputs or for creating other, more complex feature detectors. The Cascade-Correlation architecture has several advantages over existing algorithms: it learns very quickly, the network determines its own size and topology, it retains the structures it has built even if the training set changes, and it requires no back-propagation of error signals through the connections of the network.

## 1    DESCRIPTION OF CASCADE-CORRELATION

The most important problem preventing the widespread application of artificial neural networks to real-world problems is the slowness of existing learning algorithms such as back-propagation (or "backprop"). One factor contributing to that slowness is what we call the *moving target problem*: because all of the weights in the network are changing at once, each hidden units sees a constantly changing environment. Instead of moving quickly to assume useful roles in the overall problem solution, the hidden units engage in a complex dance with much wasted motion. The Cascade-Correlation learning algorithm was developed in an attempt to solve that problem. In the problems we have examined, it learns much faster than back-propagation and solves some other problems as well.

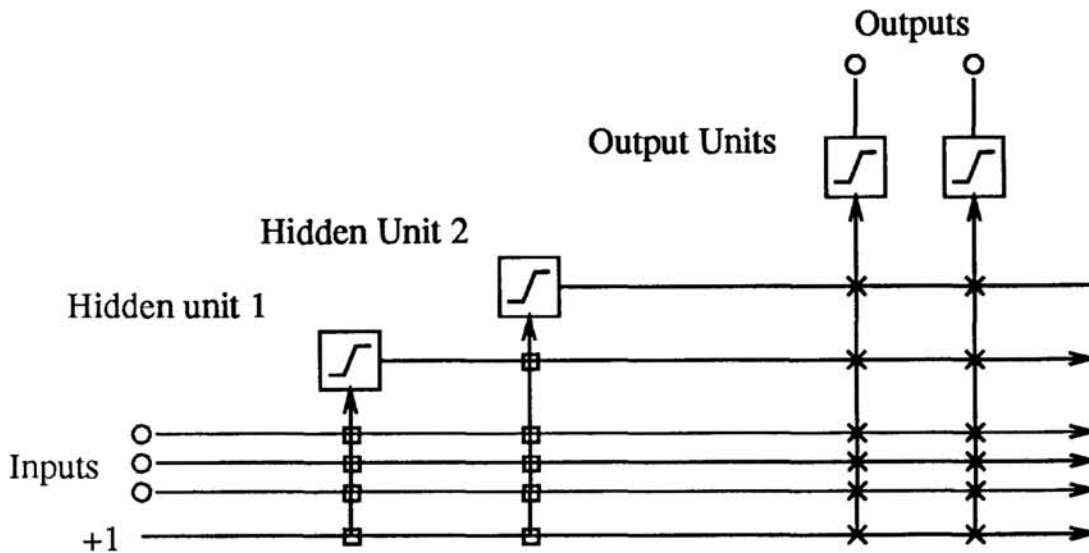

**Figure 1:** The Cascade architecture, after two hidden units have been added. The vertical lines sum all incoming activation. Boxed connections are frozen, X connections are trained repeatedly.

Cascade-Correlation combines two key ideas: The first is the *cascade architecture*, in which hidden units are added to the network one at a time and do not change after they have been added. The second is the learning algorithm, which creates and installs the new hidden units. For each new hidden unit, we attempt to maximize the magnitude of the *correlation* between the new unit's output and the residual error signal we are trying to eliminate.

The cascade architecture is illustrated in Figure 1. It begins with some inputs and one or more output units, but with no hidden units. The number of inputs and outputs is dictated by the problem and by the I/O representation the experimenter has chosen. Every input is connected to every output unit by a connection with an adjustable weight. There is also a *bias* input, permanently set to +1.

The output units may just produce a linear sum of their weighted inputs, or they may employ some non-linear activation function. In the experiments we have run so far, we use a symmetric sigmoidal activation function (hyperbolic tangent) whose output range is -1.0 to +1.0. For problems in which a precise analog output is desired, instead of a binary classification, linear output units might be the best choice, but we have not yet studied any problems of this kind.

We add hidden units to the network one by one. Each new hidden unit receives a connection from each of the network's original inputs and also from every pre-existing hidden unit. The hidden unit's input weights are frozen at the time the unit is added to the net; only the output connections are trained repeatedly. Each new unit therefore adds

a new one-unit "layer" to the network, unless some of its incoming weights happen to be zero. This leads to the creation of very powerful high-order feature detectors; it also may lead to very deep networks and high fan-in to the hidden units. There are a number of possible strategies for minimizing the network depth and fan-in as new units are added, but we have not yet explored these strategies.

The learning algorithm begins with no hidden units. The direct input-output connections are trained as well as possible over the entire training set. With no need to back-propagate through hidden units, we can use the Widrow-Hoff or "delta" rule, the Perceptron learning algorithm, or any of the other well-known learning algorithms for single-layer networks. In our simulations, we use Fahlman's "quickprop" algorithm [Fahlman, 1988] to train the output weights. With no hidden units, this acts essentially like the delta rule, except that it converges much faster.

At some point, this training will approach an asymptote. When no significant error reduction has occurred after a certain number of training cycles (controlled by a "patience" parameter set by the operator), we run the network one last time over the entire training set to measure the error. If we are satisfied with the network's performance, we stop; if not, we attempt to reduce the residual errors further by adding a new hidden unit to the network. The unit-creation algorithm is described below. The new unit is added to the net, its input weights are frozen, and all the output weights are once again trained using quickprop. This cycle repeats until the error is acceptably small (or until we give up).

To create a new hidden unit, we begin with a *candidate unit* that receives trainable input connections from all of the network's external inputs and from all pre-existing hidden units. The output of this candidate unit is not yet connected to the active network. We run a number of passes over the examples of the training set, adjusting the candidate unit's input weights after each pass. The goal of this adjustment is to maximize $S$, the sum over all output units $o$ of the magnitude of the correlation (or, more precisely, the covariance) between $V$, the candidate unit's value, and $E_o$, the residual output error observed at unit $o$. We define $S$ as

$$S = \sum_o \left| \sum_p (V_p - \overline{V})(E_{p,o} - \overline{E_o}) \right|$$

where $o$ is the network output at which the error is measured and $p$ is the training pattern. The quantities $\overline{V}$ and $\overline{E_o}$ are the values of $V$ and $E_o$ averaged over all patterns.

In order to maximize $S$, we must compute $\partial S / \partial w_i$, the partial derivative of $S$ with respect to each of the candidate unit's incoming weights, $w_i$. In a manner very similar to the derivation of the back-propagation rule in [Rumelhart, 1986], we can expand and differentiate the formula for $S$ to get

$$\partial S / \partial w_i = \sum_{p,o} \sigma_o(E_{p,o} - \overline{E_o}) f_p' I_{i,p}$$

where $\sigma_o$ is the sign of the correlation between the candidate's value and output $o$, $f_p'$ is

the derivative for pattern $p$ of the candidate unit's activation function with respect to the sum of its inputs, and $I_{i,p}$ is the input the candidate unit receives from unit $i$ for pattern $p$.

After computing $\partial S / \partial w_i$ for each incoming connection, we can perform a gradient ascent to maximize $S$. Once again we are training only a single layer of weights. Once again we use the quickprop update rule for faster convergence. When $S$ stops improving, we install the new candidate as a unit in the active network, freeze its input weights, and continue the cycle as described above.

Because of the absolute value in the formula for $S$, a candidate unit cares only about the *magnitude* of its correlation with the error at a given output, and not about the sign of the correlation. As a rule, if a hidden unit correlates positively with the error at a given unit, it will develop a negative connection weight to that unit, attempting to cancel some of the error; if the correlation is negative, the output weight will be positive. Since a unit's weights to different outputs may be of mixed sign, a unit can sometimes serve two purposes by developing a positive correlation with the error at one output and a negative correlation with the error at another.

Instead of a single candidate unit, it is possible to use a *pool* of candidate units, each with a different set of random initial weights. All receive the same input signals and see the same residual error for each pattern and each output. Because they do not interact with one another or affect the active network during training, all of these candidate units can be trained in parallel; whenever we decide that no further progress is being made, we install the candidate whose correlation score is the best. The use of this pool of candidates is beneficial in two ways: it greatly reduces the chance that a useless unit will be permanently installed because an individual candidate got stuck during training, and (on a parallel machine) it can speed up the training because many parts of weight-space can be explored simultaneously.

The hidden and candidate units may all be of the same type, for example with a sigmoid activation function. Alternatively, we might create a pool of candidate units with a mixture of nonlinear activation functions—some sigmoid, some Gaussian, some with radial activation functions, and so on—and let them compete to be chosen for addition to the active network. To date, we have explored the all-sigmoid and all-Gaussian cases, but we do not yet have extensive simulation data on networks with mixed unit-types.

One final note on the implementation of this algorithm: While the weights in the output layer are being trained, the other weights in the active network are frozen. While the candidate weights are being trained, none of the weights in the active network are changed. In a machine with plenty of memory, it is possible to record the unit-values and the output errors for an entire epoch, and then to use these cached values repeatedly during training, rather than recomputing them repeatedly for each training case. This can result in a tremendous speedup as the active network grows large.

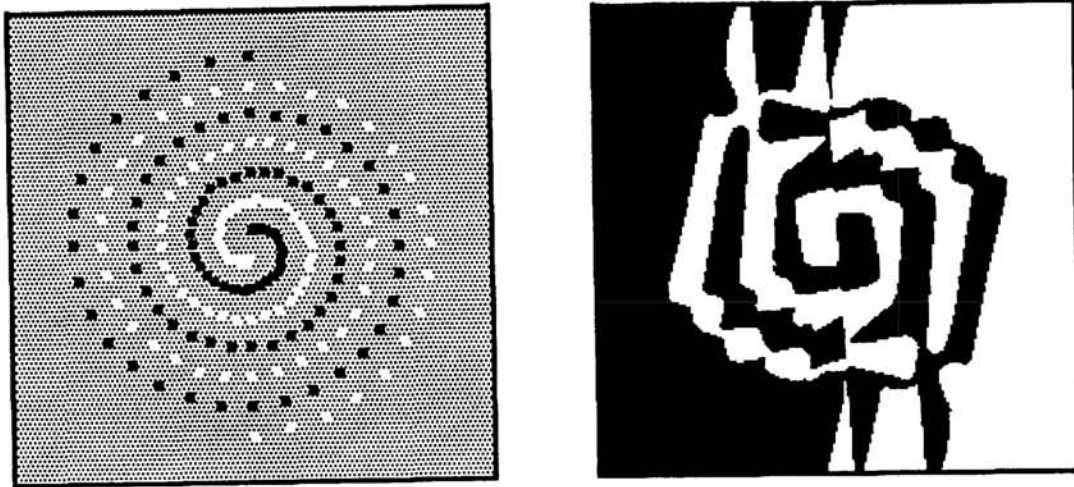

**Figure 2:** Training points for the two-spirals problem, and output pattern for one network trained with Cascade-Correlation.

## 2   BENCHMARK RESULTS

### 2.1   THE TWO-SPIRALS PROBLEM

The "two-spirals" benchmark was chosen as the primary benchmark for this study because it is an extremely hard problem for algorithms of the back-propagation family to solve. It was first proposed by Alexis Wieland of MITRE Corp. The net has two continuous-valued inputs and a single output. The training set consists of 194 X-Y values, half of which are to produce a +1 output and half a -1 output. These training points are arranged in two interlocking spirals that go around the origin three times, as shown in Figure 2a. The goal is to develop a feed-forward network with sigmoid units that properly classifies all 194 training cases. Some hidden units are obviously needed, since a single linear separator cannot divide two sets twisted together in this way.

Wieland (unpublished) reported that a modified version of backprop in use at MITRE required 150,000 to 200,000 epochs to solve this problem, and that they had never obtained a solution using standard backprop. Lang and Witbrock [Lang, 1988] tried the problem using a 2-5-5-5-1 network (three hidden layers of five units each). Their network was unusual in that it provided "shortcut" connections: each unit received incoming connections from every unit in *every* earlier layer, not just from the immediately preceding layer. With this architecture, standard backprop was able to solve the problem in 20,000 epochs, backprop with a modified error function required 12,000 epochs, and quickprop required 8000. This was the best two-spirals performance reported to date. Lang and Witbrock also report obtaining a solution with a 2-5-5-1 net (only ten hidden units in all), but the solution required 60,000 quickprop epochs.

We ran the problem 100 times with the Cascade-Correlation algorithm using a sigmoidal activation function for both the output and hidden units and a pool of 8 candidate units. All trials were successful, requiring 1700 epochs on the average. (This number counts

both the epochs used to train output weights and the epochs used to train candidate units.) The number of hidden units built into the net varied from 12 to 19, with an average of 15.2 and a median of 15. Here is a histogram of the number of hidden units created:

| Hidden Units | Number of Trials | |
|---|---|---|
| 12 | 4 | #### |
| 13 | 9 | ######### |
| 14 | 24 | ######################## |
| 15 | 19 | ################### |
| 16 | 24 | ######################## |
| 17 | 13 | ############# |
| 18 | 5 | ##### |
| 19 | 2 | ## |

In terms of training epochs, Cascade-Correlation beats quickprop by a factor of 5 and standard backprop by a factor of 10, while building a network of about the same complexity (15 hidden units). In terms of actual computation on a serial machine, however, the speedup is much greater than these numbers suggest. In backprop and quickprop, each training case requires a forward and a backward pass through all the connections in the network; Cascade-Correlation requires only a forward pass. In addition, many of the Cascade-Correlation epochs are run while the network is much smaller than its final size. Finally, the cacheing strategy described above makes it possible to avoid re-computing the unit values for parts of the network that are not changing.

Suppose that instead of epochs, we measure learning time in *connection crossings*, defined as the number of multiply-accumulate steps necessary to propagate activation values forward through the network and error values backward. This measure leaves out some computational steps, but it is a more accurate measure of computational complexity than comparing epochs of different sizes or comparing runtimes on different machines. The Lang and Witbrock result of 20,000 backprop epochs requires about 1.1 billion connection crossings. Their solution using 8000 quickprop epochs on the same network requires about 438 million crossings. An average Cascade-Correlation run with a pool of 8 candidate units requires about 19 million crossings—a 23-fold speedup over quickprop and a 50-fold speedup over standard backprop. With a smaller pool of candidate units the speedup (on a serial machine) would be even greater, but the resulting networks might be somewhat larger.

Figure 2b shows the output of a 12-hidden-unit network built by Cascade-Correlation as the input is scanned over the X-Y field. This network properly classifies all 194 training points. We can see that it interpolates smoothly for about the first 1.5 turns of the spiral, but becomes a bit lumpy farther out, where the training points are farther apart. This "receptive field" diagram is similar to that obtained by Lang and Witbrock using backprop, but is somewhat smoother.

## 2.2    N-INPUT PARITY

Since parity has been a popular benchmark among other researchers, we ran Cascade-Correlation on N-input parity problems with N ranging from 2 to 8. The best results were obtained with a sigmoid output unit and hidden units whose output is a Gaussian function of the sum of weighted inputs. Based on five trials for each value of N, our results were as follows:

| N | Cases | Hidden Units | Average Epochs |
|---|-------|--------------|----------------|
| 2 | 4 | 1 | 24 |
| 3 | 8 | 1 | 32 |
| 4 | 16 | 2 | 66 |
| 5 | 32 | 2–3 | 142 |
| 6 | 64 | 3 | 161 |
| 7 | 128 | 4–5 | 292 |
| 8 | 256 | 4–5 | 357 |

For a rough comparison, Tesauro and Janssens [Tesauro, 1988] report that standard back-prop takes about 2000 epochs for 8-input parity. In their study, they used $2N$ hidden units. Cascade-Correlation can solve the problem with fewer than $N$ hidden units because it uses short-cut connections.

As a test of generalization, we ran a few trials of Cascade-Correlation on the 10-input parity problem, training on either 50% or 25% of the 1024 patterns and testing on the rest. The number of hidden units built varied from 4 to 7 and training time varied from 276 epochs to 551. When trained on half of the patterns, performance on the test set averaged 96% correct; when trained on one quarter of the patterns, test-set performance averaged 90% correct. Note that the nearest neighbor algorithm would get almost all of the test-set cases wrong.

## 3    DISCUSSION

We believe that that Cascade-Correlation algorithm offers the following advantages over network learning algorithms currently in use:

- There is no need to guess the size, depth, and connectivity pattern of the network in advance. A reasonably small (though not optimal) net is built automatically, perhaps with a mixture of unit-types.

- Cascade-Correlation learns fast. In backprop, the hidden units engage in a complex dance before they settle into distinct useful roles; in Cascade-Correlation, each unit sees a fixed problem and can move decisively to solve that problem. For the problems we have investigated to date, the learning time in epochs grows roughly as $N \log N$, where $N$ is the number of hidden units ultimately needed to solve the problem.

- Cascade-Correlation can build deep nets (high-order feature detectors) without the dramatic slowdown we see in deep back-propagation networks.

- Cascade-Correlation is useful for *incremental learning*, in which new information is added to an already-trained net. Once built, a feature detector is never cannibalized. It is available from that time on for producing outputs or more complex features.

- At any given time, we train only one layer of weights in the network. The rest of the network is constant, so results can be cached.

- There is never any need to propagate error signals backwards through network connections. A single residual error signal can be broadcast to all candidates. The weighted connections transmit signals in only one direction, eliminating one difference between these networks and biological synapses.

- The candidate units do not interact, except to pick a winner. Each candidate sees the same inputs and error signals. This limited communication makes the architecture attractive for parallel implementation.

## 4   RELATION TO OTHER WORK

The principal differences between Cascade-Correlation and older learning architectures are the dynamic creation of hidden units, the way we stack the new units in multiple layers (with a fixed output layer), the freezing of units as we add them to the net, and the way we train new units by hill-climbing to maximize the unit's correlation with the residual error. The most interesting discovery is that by training one unit at a time instead of training the whole network at once, we can speed up the learning process considerably, while still creating a reasonably small net that generalizes well.

A number of researchers [Ash, 1989,Moody, 1989] have investigated networks that add new units or receptive fields within a single layer in the course of learning. While single-layer systems are well-suited for some problems, these systems are incapable of creating higher-order feature detectors that combine the outputs of existing units. The idea of building feature detectors and then freezing them was inspired in part by the work of Waibel on modular networks [Waibel, 1989], but in his model the structure of the sub-networks must be fixed before learning begins.

We know of only a few attempts to build up multi-layer networks as the learning progresses. Our decision to look at models in which each unit can see all pre-existing units was inspired to some extent by work on progressively deepening threshold-logic models by Merrick Furst and Jeff Jackson at Carnegie Mellon. (They are not actively pursuing this line at present.) Gallant [Gallant, 1986] briefly mentions a progressively deepening perceptron model (his "inverted pyramid" model) in which units are frozen after being installed. However, he has concentrated most of his research effort on models in which new hidden units are generated at random rather than by a deliberate training process. The SONN model of Tenorio and Lee [Tenorio, 1989] builds a multiple-layer topology

to suit the problem at hand. Their algorithm places new two-input units at randomly selected locations, using a simulated annealing search to keep only the most useful ones—a very different approach from ours.

**Acknowledgments**

We would like to thank Merrick Furst, Paul Gleichauf, and David Touretzky for asking good questions that helped to shape this work. This research was sponsored in part by the National Science Foundation (Contract EET-8716324) and in part by the Defense Advanced Research Projects Agency (Contract F33615-87-C-1499).

# References

[Ash, 1989]          Ash, T. (1989) "Dynamic Node Creation in Back-Propagation Networks", Technical Report 8901, Institute for Cognitive Science, University of California, San Diego.

[Fahlman, 1988]      Fahlman, S. E. (1988) "Faster-Learning Variations on Back-Propagation: An Empirical Study" in *Proceedings of the 1988 Connectionist Models Summer School*, Morgan Kaufmann.

[Gallant, 1986]      Gallant, S. I. (1986) "Three Constructive Algorithms for Network Learning" in *Proceedings, 8th Annual Conference of the Cognitive Science Society.*

[Lang, 1988]         Lang, K. J. and Witbrock, M. J. (1988) "Learning to Tell Two Spirals Apart" in *Proceedings of the 1988 Connectionist Models Summer School*, Morgan Kaufmann.

[Moody, 1989]        Moody, J. (1989) "Fast Learning in Multi-Resolution Hierarchies" in D. S. Touretzky (ed.), *Advances in Neural Information Processing Systems 1*, Morgan Kaufmann.

[Rumelhart, 1986]    Rumelhart, D. E., Hinton, G. E., and Williams, R. J. (1986) "Learning Internal Representations by Error Propagation" in Rumelhart, D. E. and McClelland, J. L.,*Parallel Distributed Processing: Explorations in the Microstructure of Cognition*, MIT Press.

[Tenorio, 1989]      Tenorio, M. F., and Lee, W. T. (1989) "Self-Organizing Neural Nets for the Identification Problem" in D. S. Touretzky (ed.), *Advances in Neural Information Processing Systems 1*, Morgan Kaufmann.

[Tesauro, 1988]      Tesauro, G. and Janssens, B. (1988) "Scaling Relations in Back-Propagation Learning" in *Complex Systems 2* 39-44.

[Waibel, 1989]       Waibel, A. (1989) "Consonant Recognition by Modular Construction of Large Phonemic Time-Delay Neural Networks" in D. S. Touretzky (ed.), *Advances in Neural Information Processing Systems 1*, Morgan Kaufmann.